# Reinforcement Learning based on On-line EM Algorithm

**Masa-aki Sato** †
†ATR Human Information Processing Research Laboratories
Seika, Kyoto 619-0288, Japan      masaaki@hip.atr.co.jp

**Shin Ishii** ‡†
‡Nara Institute of Science and Technology
Ikoma, Nara 630-0101, Japan      ishii@is.aist-nara.ac.jp

## Abstract

In this article, we propose a new reinforcement learning (RL) method based on an actor-critic architecture. The actor and the critic are approximated by Normalized Gaussian Networks (NGnet), which are networks of local linear regression units. The NGnet is trained by the on-line EM algorithm proposed in our previous paper. We apply our RL method to the task of swinging-up and stabilizing a single pendulum and the task of balancing a double pendulum near the upright position. The experimental results show that our RL method can be applied to optimal control problems having continuous state/action spaces and that the method achieves good control with a small number of trial-and-errors.

## 1   INTRODUCTION

Reinforcement learning (RL) methods (Barto et al., 1990) have been successfully applied to various Markov decision problems having finite state/action spaces, such as the backgammon game (Tesauro, 1992) and a complex task in a dynamic environment (Lin, 1992). On the other hand, applications to continuous state/action problems (Werbos, 1990; Doya, 1996; Sofge & White, 1992) are much more difficult than the finite state/action cases. Good function approximation methods and fast learning algorithms are crucial for successful applications.

In this article, we propose a new RL method that has the above-mentioned two features. This method is based on an actor-critic architecture (Barto et al., 1983), although the detailed implementations of the actor and the critic are quite differ-

ent from those in the original actor-critic model. The actor and the critic in our method estimate a policy and a Q-function, respectively, and are approximated by Normalized Gaussian Networks (NGnet) (Moody & Darken, 1989). The NGnet is a network of local linear regression units. The model softly partitions the input space by using normalized Gaussian functions, and each local unit linearly approximates the output within its partition. As pointed out by Sutton (1996), local models such as the NGnet are more suitable than global models such as multi-layered percep-trons, for avoiding serious learning interference in on-line RL processes. The NGnet is trained by the on-line EM algorithm proposed in our previous paper (Sato & Ishii, 1998). It was shown that this on-line EM algorithm is faster than a gradient descent algorithm. In the on-line EM algorithm, the positions of the local units can be adjusted according to the input and output data distribution. Moreover, unit creation and unit deletion are performed according to the data distribution. Therefore, the model can be adapted to dynamic environments in which the input and output data distribution changes with time (Sato & Ishii, 1998).

We have applied the new RL method to optimal control problems for deterministic nonlinear dynamical systems. The first experiment is the task of swinging-up and stabilizing a single pendulum with a limited torque (Doya, 1996). The second experiment is the task of balancing a double pendulum where a torque is applied only to the first pendulum. Our RL method based on the on-line EM algorithm demonstrated good performances in these experiments.

## 2 NGNET AND ON-LINE EM ALGORITHM

In this section, we review the on-line EM algorithm for the NGnet proposed in our previous paper (Sato & Ishii, 1998). The NGnet (Moody & Darken, 1989), which transforms an $N$-dimensional input vector $x$ to a $D$-dimensional output vector $y$, is defined by the following equations.

$$y = \sum_{i=1}^{M} \left( \frac{G_i(x)}{\sum_{j=1}^{M} G_j(x)} \right) (W_i x + b_i) \tag{1a}$$

$$G_i(x) \equiv (2\pi)^{-N/2} |\Sigma_i|^{-1/2} \exp\left[ -\frac{1}{2}(x - \mu_i)' \Sigma_i^{-1}(x - \mu_i) \right]. \tag{1b}$$

$M$ denotes the number of units, and the prime (') denotes a transpose. $G_i(x)$ is an $N$-dimensional Gaussian function, which has an $N$-dimensional center $\mu_i$ and an $(N \times N)$-dimensional covariance matrix $\Sigma_i$. $W_i$ and $b_i$ are a $(D \times N)$-dimensional lin-ear regression matrix and a $D$-dimensional bias vector, respectively. Subsequently, we use notations $\tilde{W}_i \equiv (W_i, b_i)$ and $\tilde{x}' \equiv (x', 1)$.

The NGnet can be interpreted as a stochastic model, in which a pair of an input and an output, $(x, y)$, is a stochastic event. For each event, a unit index $i \in \{1, ..., M\}$ is assumed to be selected, which is regarded as a hidden variable. The stochastic model is defined by the probability distribution for a triplet $(x, y, i)$, which is called a complete event:

$$P(x, y, i | \theta) = (2\pi)^{-(D+N)/2} \sigma_i^{-D} |\Sigma_i|^{-1/2} M^{-1} \tag{2}$$

$$\times \exp\left[ -\frac{1}{2}(x - \mu_i)' \Sigma_i^{-1}(x - \mu_i) - \frac{1}{2\sigma_i^2}(y - \tilde{W}_i \tilde{x})^2 \right].$$

Here, $\theta \equiv \{\mu_i, \Sigma_i, \sigma_i^2, \tilde{W}_i \mid i = 1, ..., M\}$ is a set of model parameters. We can easily prove that the expectation value of the output $y$ for a given input $x$, i.e., $E[y|x] \equiv$

$\int yP(y|x,\theta)dy$, is identical to equation (1). Namely, the probability distribution (2) provides a stochastic model for the NGnet.

From a set of $T$ events (observed data) $(X,Y) \equiv \{(x(t),y(t)) \mid t = 1, ..., T\}$, the model parameter $\theta$ of the stochastic model (2) can be determined by the maximum likelihood estimation method, in particular, by the EM algorithm (Dempster et al., 1977). The EM algorithm repeats the following E- and M-steps.

E (Estimation) step:      Let $\bar{\theta}$ be the present estimator. By using $\bar{\theta}$, the posterior probability that the $i$-th unit is selected for $(x(t), y(t))$ is given as

$$P(i|x(t),y(t),\bar{\theta}) = P(x(t),y(t),i|\bar{\theta})/\sum_{j=1}^{M} P(x(t),y(t),j|\bar{\theta}). \qquad (3)$$

M (Maximization) step:      Using the posterior probability (3), the expected log-likelihood $L(\theta|\bar{\theta}, X, Y)$ for the complete events is defined by

$$L(\theta|\bar{\theta}, X, Y) = \sum_{t=1}^{T}\sum_{i=1}^{M} P(i|x(t),y(t),\bar{\theta})\log P(x(t),y(t),i|\theta). \qquad (4)$$

Since an increase of $L(\theta|\bar{\theta}, X, Y)$ implies an increase of the log-likelihood for the observed data $(X, Y)$ (Dempster et al., 1977), $L(\theta|\bar{\theta}, X, Y)$ is maximized with respect to $\theta$. A solution of the necessity condition $\partial L/\partial \theta = 0$ is given by (Xu et al., 1995) .

$$\mu_i = \langle x \rangle_i(T)/\langle 1 \rangle_i(T) \qquad (5a)$$

$$\Sigma_i^{-1} = [\langle xx' \rangle_i(T)/\langle 1 \rangle_i(T) - \mu_i(T)\mu_i'(T)]^{-1} \qquad (5b)$$

$$\tilde{W}_i = \langle y\tilde{x}' \rangle_i(T)[\langle \tilde{x}\tilde{x}' \rangle_i(T)]^{-1} \qquad (5c)$$

$$\sigma_i^2 = \frac{1}{D}\left[\langle |y^2| \rangle_i(T) - \text{Tr}\left(\tilde{W}_i\langle \tilde{x}y' \rangle_i(T)\right)\right]/\langle 1 \rangle_i(T), \qquad (5d)$$

where $\langle \cdot \rangle_i$ denotes a weighted mean with respect to the posterior probability (3) and it is defined by

$$\langle f(x,y) \rangle_i(T) \equiv \frac{1}{T}\sum_{t=1}^{T} f(x(t),y(t))P(i|x(t),y(t),\bar{\theta}). \qquad (6)$$

The EM algorithm introduced above is based on batch learning (Xu et al., 1995), namely, the parameters are updated after seeing all of the observed data. We introduce here an on-line version (Sato & Ishii, 1998) of the EM algorithm. Let $\theta(t)$ be the estimator after the $t$-th observed data $(x(t), y(t))$. In this on-line EM algorithm, the weighted mean (6) is replaced by

$$\ll f(x,y) \gg_i (T) \equiv \eta(T)\sum_{t=1}^{T}(\prod_{s=t+1}^{T} \lambda(s))f(x(t),y(t))P(i|x(t),y(t),\theta(t-1)). \qquad (7)$$

The parameter $\lambda(t) \in [0, 1]$ is a discount factor, which is introduced for forgetting the effect of earlier inaccurate estimator. $\eta(T) \equiv (\sum_{t=1}^{T}(\prod_{s=t+1}^{T} \lambda(s)))^{-1}$ is a normalization coefficient and it is iteratively calculated by $\eta(t) = (1 + \lambda(t)/\eta(t-1))^{-1}$. The modified weighted mean $\ll \cdot \gg_i$ can be obtained by the step-wise equation:

$$\ll f(x,y) \gg_i (t) = \ll f(x,y) \gg_i (t-1) \qquad (8)$$
$$+\eta(t)\left[f(x(t),y(t))P_i(t)- \ll f(x,y) \gg_i (t-1)\right],$$

where $P_i(t) \equiv P(i|x(t), y(t), \theta(t-1))$. Using the modified weighted mean, the new parameters are obtained by the following equations.

$$\tilde{\Lambda}_i(t) = \frac{1}{1 - \eta(t)} \left[ \tilde{\Lambda}_i(t-1) - \frac{P_i(t)\tilde{\Lambda}_i(t-1)\tilde{x}(t)\tilde{x}'(t)\tilde{\Lambda}_i(t-1)}{(1/\eta(t)-1) + P_i(t)\tilde{x}'(t)\tilde{\Lambda}_i(t-1)\tilde{x}(t)} \right] \quad (9a)$$

$$\mu_i(t) = \ll x \gg_i (t)/ \ll 1 \gg_i (t) \quad (9b)$$

$$\tilde{W}_i(t) = \tilde{W}_i(t-1) + \eta(t)P_i(t)(y(t) - \tilde{W}_i(t-1)\tilde{x}(t))\tilde{x}'(t)\tilde{\Lambda}_i(t) \quad (9c)$$

$$\sigma_i^2(t) = \frac{1}{D} \left[ \ll |y|^2 \gg_i (t) - \text{Tr} \left( \tilde{W}_i(t) \ll \tilde{x}y' \gg_i (t) \right) \right] / \ll 1 \gg_i (t), \quad (9d)$$

where $\tilde{\Lambda}_i(t) \equiv [\ll \tilde{x}\tilde{x}' \gg_i]^{-1}$. $\Sigma_i^{-1}(t)$ can be obtained from the following relation with $\tilde{\Lambda}_i(t)$.

$$\tilde{\Lambda}_i(t) \ll 1 \gg_i (t) = \begin{pmatrix} \Sigma_i^{-1}(t) & -\Sigma_i^{-1}(t)\mu_i(t) \\ -\mu_i'(t)\Sigma_i^{-1}(t) & 1 + \mu_i'(t)\Sigma_i^{-1}(t)\mu_i(t) \end{pmatrix}. \quad (10)$$

It can be proved that this on-line EM algorithm is equivalent to the stochastic approximation for finding the maximum likelihood estimator, if the time course of the discount factor $\lambda(t)$ is given by

$$\lambda(t) \xrightarrow{t \to \infty} 1 - (1-a)/(at + b), \quad (11)$$

where $a$ $(1 > a > 0)$ and $b$ are constants (Sato & Ishii, 1998).

We also employ dynamic unit manipulation mechanisms in order to efficiently allocate the units (Sato & Ishii, 1998). The probability $P(x(t), y(t), i \mid \theta(t-1))$ indicates how probable the $i$-th unit produces the datum $(x(t), y(t))$ with the present parameter $\theta(t-1)$. If the probability for every unit is less than some threshold value, a new unit is produced to account for the new datum. The weighted mean $\ll 1 \gg_i (t)$ indicates how much the $i$-th unit has been used to account for the data until $t$. If the mean becomes less than some threshold value, this unit is deleted.

In order to deal with a singular input distribution, a regularization for $\Sigma_i^{-1}(t)$ is introduced as follows.

$$\Sigma_i^{-1}(t) = [(\ll xx' \gg_i (t) - \mu_i(t)\mu_i'(t) \ll 1 \gg_i (t) \quad (12a)$$

$$+ \alpha \ll \Delta_i^2 \gg_i (t)I_N) / \ll 1 \gg_i (t)]^{-1}$$

$$\ll \Delta_i^2 \gg_i (t) = (\ll |x|^2 \gg_i (t) - |\mu_i(t)|^2 \ll 1 \gg_i (t)) /N, \quad (12b)$$

where $I_N$ is the $(N \times N)$-dimensional identity matrix and $\alpha$ is a small constant. The corresponding $\tilde{\Lambda}_i(t)$ can be calculated in an on-line manner using a similar equation to (9a) (Sato & Ishii, 1998).

## 3  REINFORCEMENT LEARNING

In this section, we propose a new RL method based on the on-line EM algorithm described in the previous section. In the following, we consider optimal control problems for deterministic nonlinear dynamical systems having continuous state/action spaces. It is assumed that there is no knowledge of the controlled system. An actor-critic architecture (Barto et al.,1983) is used for the learning system. In the original actor-critic model, the actor and the critic approximated the probability of each action and the value function, respectively, and were trained by using the TD-error. The actor and the critic in our RL method are different from those in the original model as explained later.

For the current state, $x_c(t)$, of the controlled system, the actor outputs a control signal (action) $u(t)$, which is given by the policy function $\Omega(\cdot)$, i.e., $u(t) = \Omega(x_c(t))$. The controlled system changes its state to $x_c(t+1)$ after receiving the control signal $u(t)$. Subsequently, a reward $r(x_c(t), u(t))$ is given to the learning system. The objective of the learning system is to find the optimal policy function that maximizes the discounted future return defined by

$$V(x_c) \equiv \sum_{t=0}^{\infty} \gamma^t r(x_c(t), \Omega(x_c(t)))\big|_{x_c(0) = x_c}, \tag{13}$$

where $0 < \gamma < 1$ is a discount factor. $V(x_c)$, which is called the value function, is defined for the current policy function $\Omega(\cdot)$ employed by the actor. The Q-function is defined by

$$Q(x_c, u) = \gamma V(x_c(t+1)) + r(x_c, u), \tag{14}$$

where $x_c(t) = x_c$ and $u(t) = u$ are assumed. The value function can be obtained from the Q-function:

$$V(x_c) = Q(x_c, \Omega(x_c)). \tag{15}$$

The Q-function should satisfy the consistency condition

$$Q(x_c(t), u(t)) = \gamma Q(x_c(t+1), \Omega(x_c(t+1))) + r(x_c(t), u(t)). \tag{16}$$

In our RL method, the policy function and the Q-function are approximated by the NGnets, which are called the actor-network and the critic-network, respectively. In the learning phase, a stochastic actor is necessary in order to explore a better policy. For this purpose, we employ a stochastic model defined by (2), corresponding to the actor-network. A stochastic action is generated in the following way. A unit index $i$ is selected randomly according to the conditional probability $P(i|x_c)$ for a given state $x_c$. Subsequently, an action $u$ is generated randomly according to the conditional probability $P(u|x_c, i)$ for a given $x_c$ and the selected $i$. The value function can be defined for either the stochastic policy or the deterministic policy. Since the controlled system is deterministic, we use the value function defined for the deterministic policy which is given by the actor-network.

The learning process proceeds as follows. For the current state $x_c(t)$, a stochastic action $u(t)$ is generated by the stochastic model corresponding to the current actor-network. At the next time step, the learning system gets the next state $x_c(t+1)$ and the reward $r(x_c(t), u(t))$. The critic-network is trained by the on-line EM algorithm. The input to the critic-network is $(x_c(t), u(t))$. The target output is given by the right hand side of (16), where the Q-function and the deterministic policy function $\Omega(\cdot)$ are calculated using the current critic-network and the current actor-network, respectively. The actor-network is also trained by the on-line EM algorithm. The input to the actor-network is $x_c(t)$. The target output is given by using the gradient of the critic-network (Sofge & White, 1992):

$$u_{target} = \Omega(x_c(t)) + \epsilon \frac{\partial Q}{\partial u}(x_c(t), \Omega(x_c(t))), \tag{17}$$

where the Q-function and the deterministic policy function $\Omega(\cdot)$ are calculated using the modified critic-network and the current actor-network, respectively. $\epsilon$ is a small constant. This target output gives a better action, which increases the Q-function value for the current state $x_c(t)$, than the current deterministic action $\Omega(x_c(t))$.

In the above learning scheme, the critic-network and the actor-network are updated concurrently. One can consider another learning scheme. In this scheme, the learning system tries to control the controlled system for a given period of time by using the fixed actor-network. In this period, the critic-network is trained to estimate the

Q-function for the fixed actor-network. The state trajectory in this period is saved. At the next stage, the actor-network is trained along the saved trajectory using the critic-network modified in the first stage.

## 4   EXPERIMENTS

The first experiment is the task of swinging-up and stabilizing a single pendulum with a limited torque (Doya, 1996). The state of the pendulum is represented by $x_c \equiv (\phi, \dot{\phi})$, where $\phi$ and $\dot{\phi}$ denote the angle from the upright position and the angular velocity of the pendulum, respectively. The reward $r(x_c(t), u(t))$ is assumed to be given by $\tilde{r}(x_c(t + 1))$, where

$$\tilde{r}(x_c) = \exp(-(\dot{\phi})^2/(2\nu_1^2) - \phi^2/(2\nu_2^2)). \tag{18}$$

$\nu_1$ and $\nu_2$ are constants. The reward (18) encourages the pendulum to stay high. After releasing the pendulum from a vicinity of the upright position, the control and the learning process of the actor-critic network is conducted for 7 seconds. This is a single episode. The reinforcement learning is done by repeating these episodes. After 40 episodes, the system is able to make the pendulum achieve an upright position from almost every initial state. Even from a low initial position, the system swings the pendulum several times and stabilizes it at the upright position. Figure 1 shows a control process, i.e., stroboscopic time-series of the pendulum, using the deterministic policy after training. According to our previous experiment, in which both of the actor- and critic- networks are the NGnets with fixed centers trained by the gradient descent algorithm, a good control was obtained after about 2000 episodes. Therefore, our new RL method is able to obtain a good control much faster than that based on the gradient descent algorithm.

The second experiment is the task of balancing a double pendulum near the upright position. A torque is applied only to the first pendulum. The state of the pendulum is represented by $x_c \equiv (\phi_1, \phi_2, \dot{\phi}_1, \dot{\phi}_2)$, where $\phi_1$ and $\phi_2$ are the first pendulum's angle from the upright direction and the second pendulum's angle from the first pendulum's direction, respectively. $\dot{\phi}_1(\dot{\phi}_2)$ is the angular velocity of the first (second) pendulum. The reward is given by the height of the second pendulum's end from the lowest position. After 40 episodes, the system is able to stabilize the double pendulum. Figure 2 shows the control process using the deterministic policy after training. The upper two figures show stroboscopic time-series of the pendulum. The dashed, dotted, and solid lines in the bottom figure denote $\phi_1/\pi$, $\phi_2/\pi$, and the control signal $u$ produced by the actor-network, respectively. After a transient period, the pendulum is successfully controlled to stay near the upright position.

The numbers of units in the actor- (critic-) networks after training are 50 (109) and 96 (121) for the single and double pendulum cases, respectively. The RL method using center-fixed NGnets trained by the gradient descent algorithm employed 441 $(= 21^2)$ actor units and 18,081 $(= 21^2 \times 41)$ critic units, for the single pendulum task. For the double pendulum task, this scheme did not work even when 14,641 $(= 11^4)$ actor units and 161,051 $(= 11^4 \times 11)$ critic units were prepared. The numbers of units in the NGnets trained by the on-line EM algorithm scale moderately as the input dimension increases.

## 5   CONCLUSION

In this article, we proposed a new RL method based on the on-line EM algorithm. We showed that our RL method can be applied to the task of swinging-up and

stabilizing a single pendulum and the task of balancing a double pendulum near the upright position. The number of trial-and-errors needed to achieve good control was found to be very small in the two tasks. In order to apply a RL method to continuous state/action problems, good function approximation methods and fast learning algorithms are crucial. The experimental results showed that our RL method has both features.

## References

Barto, A. G., Sutton, R. S., & Anderson, C. W. (1983). *IEEE Transactions on Systems, Man, and Cybernetics*, **13**, 834-846.

Barto, A. G., Sutton, R. S., & Watkins, C. J. C. H. (1990). *Learning and Computational Neuroscience: Foundations of Adaptive Networks* (pp. 539-602), MIT Press.

Dempster, A. P., Laird, N. M., & Rubin, D. B. (1977). *Journal of Royal Statistical Society B*, **39**, 1-22.

Doya, K. (1996). *Advances in Neural Information Processing Systems 8* (pp. 1073-1079), MIT Press.

Lin, L. J. (1992). *Machine Learning*, **8**, 293-321.

Moody, J., & Darken, C. J. (1989). *Neural Computation*, **1**, 281-294.

Sato, M., & Ishii, S. (1998). *ATR Technical Report*, **TR-H-243**, ATR.

Sofge, D. A., & White, D. A. (1992). *Handbook of Intelligent Control* (pp. 259-282), Van Nostrand Reinhold.

Sutton, R. S. (1996). *Advances in Neural Information Processing Systems 8* (pp. 1038-1044), MIT Press.

Tesauro, G. J. (1992). *Machine Learning*, **8**, 257-278.

Werbos, P. J. (1990). *Neural Networks for Control* (pp. 67-95), MIT Press.

Xu, L., Jordan, M. I., & Hinton, G. E. (1995). *Advances in Neural Information Processing Systems 7* (pp. 633-640), MIT Press.

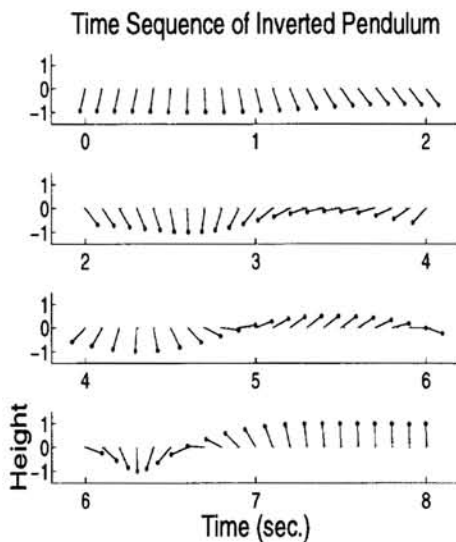

**Figure 1**

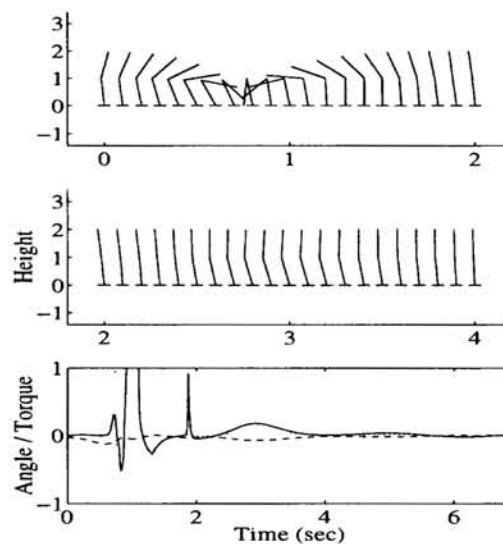

**Figure 2**